# Learning Bounds for a Generalized Family of Bayesian Posterior Distributions

**Tong Zhang**
IBM T.J. Watson Research Center
Yorktown Heights, NY 10598
tzhang@watson.ibm.com

## Abstract

In this paper we obtain convergence bounds for the concentration of Bayesian posterior distributions (around the true distribution) using a novel method that simplifies and enhances previous results. Based on the analysis, we also introduce a generalized family of Bayesian posteriors, and show that the convergence behavior of these generalized posteriors is completely determined by the local prior structure around the true distribution. This important and surprising robustness property does not hold for the standard Bayesian posterior in that it may not concentrate when there exist "bad" prior structures even at places far away from the true distribution.

## 1 Introduction

Consider a sample space $\mathcal{X}$ and a measure $\lambda$ on $\mathcal{X}$ (with respect to some $\sigma$-field). In statistical inference, the nature picks a probability measure $Q$ on $\mathcal{X}$ which is unknown. We assume that $Q$ has a density $q$ with respect to $\lambda$. In the Bayesian paradigm, the statistician considers a set of probability densities $p(\cdot|\theta)$ (with respect to $\lambda$ on $\mathcal{X}$) indexed by $\theta \in \Gamma$, and makes an assumption[1] that the true density $q$ can be represented as $p(\cdot|\theta)$ with $\theta$ randomly picked from $\Gamma$ according to a prior distribution $\pi$ on $\Gamma$. Throughout the paper, all quantities appearing in the derivations are assumed to be measurable.

Given a set of samples $X = \{X_1, \ldots, X_n\} \in \mathcal{X}^n$, where each $X_i$ independently drawn from (the unknown distribution) $Q$, the optimal Bayesian method can be derived as the optimal inference with respect to the posterior distribution. Although a Bayesian procedure is optimal only when the nature picks the same prior as the statistician (which is very unlikely), it is known that procedures with desirable properties from the frequentist point of view (such minimaxity and admissibility) are often Bayesian [6]. From a theoretical point of view, it is necessary to understand the behavior of Bayesian methods without the assumption that the nature picks the same prior as the statistician. In this respect, the most fundamental issue in Bayesian analysis is whether the Bayesian inference based on the posterior distribution will converge to the corresponding inference of the true (but

unknown) distribution when the number of observations approach infinity.

A more general question is whether the Bayesian posterior distribution will be concentrated around the true underlying distribution when the sample size is large. This is often referred to as the consistency of Bayesian posterior distribution, which is certainly the most fundamental issue for understanding the behavior of Bayesian methods. This problem has drawn considerable attention in statistics. The classical results include average consistency results such as Doob's consistency theorem and asymptotic convergence results such as the Bernstein-von Mises theorem for parametric problems. For infinite-dimensional problems, one has to choose the prior very carefully, or the Bayesian posterior may not concentrate around the true underlying distribution, which leads to inconsistency [1, 2]. In [1], the authors also gave conditions that guarantee the consistency of Bayesian posterior distributions, although convergence rates were not obtained. The convergence rates were studied in two recent works [3, 8] by using heavy machineries from the empirical process theory.

The purpose of this paper is to develop finite-sample convergence bounds for Bayesian posterior distributions using a novel approach that not only simplifies the analysis given in [3, 8], but also leads to tighter bounds. At the heart of our approach are some new posterior averaging bounds that are related to the PAC Bayes analysis appeared in some recent machine learning works. These new bounds are of independent interests (though we cannot fully explore their consequences here) since they can be used to obtain correct convergence rates for other statistical estimation problems such as least squares regression. Motivated by our learning bounds, we introduce a generalized family of Bayesian methods, and show that their convergence behavior relies only on the prior mass in a small neighborhood around the true distribution. This is rather surprising when we consider the example given in [1], which shows that for the (standard) Bayesian method, even if one puts a positive prior mass around the true distribution, one may still get an inconsistent posterior when there exist undesirable prior structures far away from the true distribution.

## 2   The regularization formulation of Bayesian posterior measure

Assume we observe $n$-samples $X = \{X_1, \ldots, X_n\} \in \mathcal{X}^n$, independently drawn from the true underlying distribution $Q$. We shall call any probability density $\hat{w}_X(\theta)$ with respect to $\pi$ that depends on the observation $X$ (and measurable on $\mathcal{X}^n \times \Gamma$) a posterior distribution. $\forall \alpha \in (0, 1]$, we define a generalized Bayesian posterior $\pi^\alpha(\cdot|X)$ with respect to $\pi$ as:

$$\pi^\alpha(\theta|X) = \frac{\prod_{i=1}^n p^\alpha(X_i|\theta)}{\int_\Gamma \prod_{i=1}^n p^\alpha(X_i|\theta) d\pi(\theta)}. \tag{1}$$

We call $\pi^\alpha$ the $\alpha$-Bayesian posterior. The standard Bayesian posterior is denoted as $\pi(\cdot|X) = \pi^1(\cdot|X)$. Given a probability density $w(\cdot)$ on $\Gamma$ with respect to $\pi$, we define the KL-divergence $\mathbf{KL}(wd\pi||d\pi)$ as:

$$\mathbf{KL}(wd\pi||d\pi) = \int_\Gamma w(\theta) \ln w(\theta) d\pi(\theta).$$

Consider a real-valued function $f(\theta)$ on $\Gamma$, we denote by $\mathbf{E}_\pi f(\theta)$ the expectation of $f(\cdot)$ with respect to $\pi$. Similarly, consider a real-valued function $\ell(x)$ on $\mathcal{X}$, we denote by $\mathbf{E}_q \ell(x)$ the expectation of $\ell(\cdot)$ with respect the true underlying distribution $q$. We also use $\mathbf{E}_X$ to denote the expectation with respect to the observation $X$.

The key starting point of our analysis is the following simple observation that relates the Bayesian posterior to the solution of an entropy regularized density (with respect to $\pi$) estimation. Under this formulation, techniques for analyzing regularized risk minimization problems, such as those recently investigated by the author, can be applied to obtain sample

complexity bound for Bayesian posterior distributions. The proof of the following regularization formulation is straight-forward, which we shall skip due to the space limitation.

**Proposition 2.1** *For any density $w$ on $\Gamma$ with respect to $\pi$, let*

$$\hat{R}_X^\alpha(w) = \alpha \frac{1}{n} \sum_{i=1}^n \mathbf{E}_\pi \, w(\theta) \ln \frac{q(X_i)}{p(X_i|\theta)} + \frac{1}{n} \mathbf{KL}(wd\pi||d\pi).$$

*Then $\hat{R}_X^\alpha(\pi^\alpha(\cdot|X)) = \inf_w \hat{R}_X^\alpha(w)$.*

The above Proposition indicates that the generalized Bayesian posterior minimizes the regularized empirical risk $\hat{R}_X^\alpha(w)$ among all possible densities $w$ with respect to the prior $\pi$. We thus only need to study the behavior of this regularized empirical risk minimization problem. One may define the true risk of $w$ by replacing the empirical expectation $\hat{\mathbf{E}}_X$ with the expectation with respect to the true underlying distribution $q$:

$$R_q^\alpha(w) = \alpha \mathbf{E}_\pi \, w(\theta) \mathbf{KL}(q||p(\cdot|\theta)) + \frac{1}{n} \mathbf{KL}(wd\pi||d\pi), \tag{2}$$

where $\mathbf{KL}(q||p) = \mathbf{E}_q \ln \frac{q(x)}{p(x)}$ is the KL-divergence between $q$ and $p$, which is always a non-negative number. This quantity is widely used to measure the closeness of two distributions $p$ and $q$. Clearly the Bayesian posterior is an approximate solution to (2) using empirical expectation. The first term of $R_q^\alpha(w)$ measures the average KL-divergence of $q$ and $p$ under the $w$-density. Since both the first term and the second term are non-negative, we know immediately that if $R_q^\alpha(w) \approx 0$, then the distribution $w$ is concentrated around $q$.

Using empirical process techniques, one would typically expect to bound $R_q^\alpha(w)$ in term of $\hat{R}_X^\alpha(w)$. Unfortunately, it does not work in our case since $\mathbf{KL}(q||p)$ is not well-defined for all $p$. This implies that as long as $w$ has non-zero concentration around a density $p$ with $\mathbf{KL}(q||p) = +\infty$, then $R_q^\alpha(w) = +\infty$. Therefore we may have $R_q^\alpha(\pi(\cdot|X)) = +\infty$ with non-zero probability even when the sample size approaches infinity.

A remedy is to consider a distance function that is always well-defined. In statistics, one often considers the $\rho$-divergence for $\rho \in (0,1)$, which is defined as:

$$D_\rho(q||p) = \frac{1}{\rho(1-\rho)} \mathbf{E}_q \left[ 1 - \left( \frac{p(x)}{q(x)} \right)^\rho \right]. \tag{3}$$

This divergence is always well-defined and $\mathbf{KL}(q||p) = \lim_{\rho \to 0} D_\rho(q||p)$. In the statistical literature, the convergence results were often specified under the Hellinger distance ($\rho = 0.5$). We would also like to mention that our learning bound derived later will become trivial when $\rho \to 0$. This is consistent with the above discussion since $R_q^\alpha$ (corresponding to $\rho = 0$) may not converge at all. However, under additional assumptions, such as the boundedness of $q/p$, $\mathbf{KL}(q||p)$ exists and can be bounded using the $\rho$-divergence $D_\rho(q||p)$.

## 3 Posterior averaging bounds under entropy regularization

The following inequality follows directly from a well-known convex duality. For example, see [5, 7] for an explanation.

**Proposition 3.1** *Assume that $f(\theta)$ is a measurable real-valued function on $\Gamma$, and $w(\theta)$ is a density with respect to $\pi$, we have*

$$\mathbf{E}_\pi \, w(\theta) f(\theta) \le \mathbf{KL}(wd\pi||d\pi) + \ln \mathbf{E}_\pi \exp(f(\theta)).$$

The main technical result which forms the basis of the paper is given by the following lemma, where we assume that $\hat{w}_X(\theta)$ is a posterior (density with respect to $\pi$ that depends on $X$ and measurable on $\mathcal{X}^n \times \Gamma$).

**Lemma 3.1** *Consider any posterior $\hat{w}_X(\theta)$. The following inequality holds for all measurable real-valued functions $L_X(\theta)$ on $\mathcal{X}^n \times \Gamma$:*

$$\mathbf{E}_X \exp\left[\mathbf{E}_\pi \hat{w}_X(\theta)(L_X(\theta) - \ln \mathbf{E}_X e^{L_X(\theta)}) - \mathbf{KL}(\hat{w}_X d\pi || d\pi)\right] \leq 1,$$

*where $\mathbf{E}_X$ is the expectation with respect to the observation $X$.*

*Proof.* From Proposition 3.1, we obtain

$$\hat{L}(X) = \mathbf{E}_\pi \hat{w}_X(\theta)(L_X(\theta) - \ln \mathbf{E}_X e^{L_X(\theta)}) - \mathbf{KL}(\hat{w}_X d\pi || d\pi)$$
$$\leq \ln \mathbf{E}_\pi \exp(L_X(\theta) - \ln \mathbf{E}_X e^{L_X(\theta)}).$$

Now applying the Fubini's theorem to interchange the order of integration, we have:

$$\mathbf{E}_X e^{\hat{L}(X)} \leq \mathbf{E}_X \mathbf{E}_\pi e^{L_X(\theta) - \ln \mathbf{E}_X \exp(L_X(\theta))} = \mathbf{E}_\pi \mathbf{E}_X e^{L_X(\theta) - \ln \mathbf{E}_X \exp(L_X(\theta))} = 1.$$
$\square$

The following corollary is a straight-forward consequence of Lemma 3.1. Note that for the Bayesian method, the loss $\ell_\theta(x)$ has a form of $\ell(p(x|\theta))$.

**Theorem 3.1 (Posterior Averaging Bounds)** *Under the notation of Lemma 3.1. Let $X = \{X_1, \ldots, X_n\}$ be $n$-samples that are independently drawn from $q$. Consider a measurable function $\ell_\theta(x) : \Gamma \times \mathcal{X} \to R$. Then $\forall t > 0$ and real number $\rho$, the following event holds with probability at least $1 - \exp(-t)$:*

$$-\mathbf{E}_\pi \hat{w}_X(\theta) \ln \mathbf{E}_q \exp(-\rho \ell_\theta(x))) \leq \frac{\rho \sum_{i=1}^n \mathbf{E}_\pi \hat{w}_X(\theta)\ell_\theta(X_i) + \mathbf{KL}(\hat{w}_X d\pi || d\pi) + t}{n}.$$

*Moreover, we have the following expected risk bound:*

$$-\mathbf{E}_X \mathbf{E}_\pi \hat{w}_X(\theta) \ln \mathbf{E}_q \exp(-\rho \ell_\theta(x))) \leq \mathbf{E}_X \frac{\rho \sum_{i=1}^n \mathbf{E}_\pi \hat{w}_X(\theta)\ell_\theta(X_i) + \mathbf{KL}(\hat{w}_X d\pi || d\pi)}{n}.$$

*Proof Sketch.* The first bound is a direct consequence of Markov inequality. The second bound can be obtained by using the fact $E_X \exp(\Delta_X) \geq \exp(E_X \Delta_X)$, which follows from the Jensen's inequality. $\square$

The above bounds are immediately applicable to Bayesian posterior distribution. The first leads to an exponential tail inequality, and the second leads to an expected risk bound.

Before analyzing Bayesian methods in detail in the next section, we shall briefly compare the above results to the so-called PAC-Bayes bounds, which can be obtained by estimating the left-hand side using the Hoeffding's inequality with an appropriately chosen $\rho$. However, in the following, we shall estimate the left-hand side using a Bernstein style bound, which is much more useful for general statistical estimation problems:

**Corollary 3.1** *Under the notation of Theorem 3.1, and assume that $\sup_{\theta, x_1, x_2} |\ell_\theta(x_1) - \ell_\theta(x_2)| \leq 1$. Then $\forall t, \rho > 0$, with probability of at least $1 - \exp(-t)$:*

$$\mathbf{E}_\pi \hat{w}_X(\theta) \mathbf{E}_q \ell_\theta(x) - \rho \phi(\rho) \mathbf{E}_\pi \hat{w}_X(\theta) \mathbf{Var}_q \ell_\theta(x) \leq \frac{1}{n} \sum_{i=1}^n \mathbf{E}_\pi \hat{w}_X(\theta)\ell_\theta(X_i)$$
$$+ \frac{\mathbf{KL}(\hat{w}_X d\pi || d\pi) + t}{\rho n},$$

*where $\phi(x) = (\exp(x) - x - 1)/x^2$ and $\mathbf{Var}_q \ell_\theta(x) = \mathbf{E}_q(\ell_\theta(x) - \mathbf{E}_q \ell_\theta(x))^2$.*

*Proof Sketch.* We follow one of the standard derivations of Bernstein inequality outlined below: it is well known that $\phi(x)$ is non-decreasing in $x$, which in turn implies that

$$\ln \mathbf{E}_q \exp(-\rho \ell_\theta(x))) \leq -\rho \mathbf{E}_q \ell_\theta(x) + \rho^2 \phi(\rho) \mathbf{E}_q (\ell_\theta(x) - \mathbf{E}_q \ell_\theta(x))^2.$$

Now applying this bound to the left hand side of Theorem 3.1, we finish the proof. $\square$

One may use the simple bound $\mathbf{Var}_q \ell_\theta(x) \leq 1/4$ and obtain[2].

$$\mathbf{E}_\pi \hat{w}_X(\theta) \mathbf{E}_q \ell_\theta(x) \leq \mathbf{E}_\pi \hat{w}_X(\theta) \sum_{i=1}^n \frac{\ell_\theta(X_i)}{n} + \left( \frac{\rho \phi(\rho)}{4} + \frac{\mathbf{KL}(\hat{w}_X d\pi || d\pi) + t}{\rho n} \right). \quad (4)$$

This inequality holds for any data-independent choice of $\rho$. However, one may easily turn it into a bound which allows $\rho$ to depend on the data using well-known techniques (see [5], for example). After we optimize $\rho$, the resulting bound becomes similar to the PAC-Bayes bound [4]. Typically the optimal $\rho$ is in the order of $\sqrt{\mathbf{KL}(\hat{w}_X d\pi || d\pi)/n}$, and hence the rate of convergence given on the right-hand side is no better than $O(\sqrt{1/n})$. However, the more interesting case is when there exists a constant $b \geq 0$ such that

$$\mathbf{E}_q (\ell_\theta(x) - \mathbf{E}_q \ell_\theta(x))^2 \leq b \mathbf{E}_q \ell_\theta(x). \quad (5)$$

This condition appears in the theoretical analysis of many statistical estimation problems, such as least squares regression, and when the loss function is non-negative (such as classification). It also appears in some analysis of maximum-likelihood estimation (log-loss), though as we shall see, log-loss can be much more directly handled in our framework using Theorem 3.1. A modified version of this condition also occurs in some recent analysis of classification problems even when the problem is not separable. We shall now assume that (5) holds. It follows from Corollary 3.1 that $\forall \rho > 0$ such that $\rho \phi(\rho) \leq 1/b$, we have

$$\mathbf{E}_\pi \hat{w}_X(\theta) \mathbf{E}_q \ell_\theta(x) \leq \frac{\rho \mathbf{E}_\pi \hat{w}_X(\theta) \sum_{i=1}^n \ell_\theta(X_i) + \mathbf{KL}(\hat{w}_X d\pi || d\pi) + t}{\rho(1 - b\rho\phi(\rho))n}. \quad (6)$$

Again the above inequality holds for any data-independent $\rho$, but we can easily turn it into a bound that allows $\rho$ to depend on $X$ using standard techniques. However we shall not list the final result here since this is not the purpose of the paper. The parameter $\rho$ can be optimized, and it is not hard to check that the resulting bound is significantly better than (4) when $\mathbf{E}_\pi \hat{w}_X(\theta) \sum_{i=1}^n \frac{\ell_\theta(X_i)}{n} \approx 0$. The "self-bounding" condition (5) holds in the theoretical analysis of many statistical estimation problems. To obtain the correct convergence behavior in such cases (including the Bayesian method which we are interested in here), inequality (4) is inadequate, and it is essential to use a Bernstein-type bound such as (6). It is also useful to point out that to analyze such problems, one actually only needs (6) with an appropriately chosen data-independent $\rho$, which will lead to the correct (minimax) rate of convergence. Note that if we choose $\rho$ to be a constant, then it is possible to achieve a bound that converges as fast as $O(1/n)$. We shall point out that in [7], a KL-divergence version of the PAC-Bayes bound was developed for the 0-1 loss using related techniques, which can lead to a rate as fast as $O(\ln n/n)$ if we make near zero errors. However, the Bernstein style bound given here is more generally applicable and is necessary for more complicated statistical estimation problems such as least squares regression.

## 4 Convergence bounds for Bayesian posterior distributions

We shall now analyze the finite sample convergence behavior of Bayesian posterior distributions using Theorem 3.1. Although the exponential tail inequality provides more detailed information, our discussion will be based on the expected risk bound for simplicity.

To analyze the Bayesian method, we let $\ell_\theta(x) = \ln(q(x)/p(x|\theta))$ in Theorem 3.1. Consider $\rho \in (0, 1)$. We also let $\hat{w}_X(\theta)$ be the Bayesian posterior $\pi^\alpha(\theta|X)$ with parameter $\alpha \in [\rho, 1]$ defined in (1). Consider an arbitrary data-independent density $w(\theta)$ with respect to $\pi$, using (3), we can obtain from Theorem 3.1 the following chain of equations:

$$\mathbf{E}_X \mathbf{E}_\pi \pi^\alpha(\theta|X) \ln \frac{1}{1 - \rho(1-\rho)D_\rho(q||p(\cdot|\theta))}$$

$$= -\mathbf{E}_X \mathbf{E}_\pi \pi^\alpha(\theta|X) \ln \mathbf{E}_q \exp\left(-\rho \ln \frac{q(x)}{p(x|\theta)}\right)$$

$$\leq \mathbf{E}_X \left[ \rho \mathbf{E}_\pi \pi^\alpha(\theta|X) \sum_{i=1}^n \frac{1}{n} \ln \frac{q(X_i)}{p(X_i|\theta)} + \frac{\mathbf{KL}(\pi^\alpha(\theta|X)d\pi||d\pi)}{n} \right]$$

$$\leq \mathbf{E}_X \left[ \alpha \mathbf{E}_\pi w(\theta) \frac{1}{n} \sum_{i=1}^n \ln \frac{q(X_i)}{p(X_i|\theta)} + \frac{\mathbf{KL}(wd\pi||d\pi)}{n} \right] + \frac{\alpha - \rho}{n} \mathbf{E}_X \sup_\theta \sum_{i=1}^n \ln \frac{p(X_i|\theta)}{q(X_i)}$$

$$= R_q^\alpha(w) + \frac{\alpha - \rho}{n} \mathbf{E}_X \sup_\theta \sum_{i=1}^n \ln \frac{p(X_i|\theta)}{q(X_i)},$$

where $R_q^\alpha(w)$ is defined in (2). Note that the first inequality follows from Theorem 3.1, and the second inequality follows from Proposition 2.1. The empirical process bound in the second term can be improved using a more precise bounding method, but we shall skip it here due to the lack of space. It is not difficult to see (also see Proposition 2.1 and Proposition 3.1) that (we skip the derivation due to the space limitation):

$$\inf_w R_q^\alpha(w) = -\frac{1}{n} \ln \mathbf{E}_\pi \exp(-\alpha n \mathbf{KL}(q||p(\cdot|\theta))).$$

Using the fact $-\ln(1-x) \geq x$ to simplify the left-hand side, we thus obtain:

$$\mathbf{E}_X \mathbf{E}_\pi \pi^\alpha(\theta|X) D_\rho(q||p(\cdot|\theta))$$
$$\leq \frac{-\ln \mathbf{E}_\pi e^{-\alpha n \mathbf{KL}(q||p(\cdot|\theta))} + (\alpha - \rho)\mathbf{E}_X \sup_\theta \sum_{i=1}^n \ln \frac{p(X_i|\theta)}{q(X_i)}}{\rho(1-\rho)n}. \tag{7}$$

In the following, we shall compare our analysis with previous results. To be consistent with the concept used in these previous studies, we shall consider the following quantity:

$$m_\pi^{\alpha,\rho}(X, \epsilon) = \mathbf{E}_\pi \pi^\alpha(\theta|X) \mathbf{1}(D_\rho(q||p(\cdot|\theta)) \geq \epsilon),$$

where $\mathbf{1}$ is the set indicator function. Intuitively $m_\pi^{\alpha,\rho}(X, \epsilon)$ is the probability mass of the $\alpha$-Bayesian posterior $\pi^\alpha(\cdot|X)$ in the region of $p(\cdot|\theta)$ that is at least $\epsilon$-distance away from $q$ in $D_\rho$-divergence. Using Markov inequality, we immediately obtain from (7) the following bound for $m_\epsilon^{\alpha,\rho}(X)$:

$$\mathbf{E}_X m_\pi^{\alpha,\rho}(X, \epsilon) \leq \frac{-\ln \mathbf{E}_\pi e^{-\alpha n \mathbf{KL}(q||p(\cdot|\theta))} + (\alpha - \rho)\mathbf{E}_X \sup_\theta \sum_{i=1}^n \ln \frac{p(X_i|\theta)}{q(X_i)}}{\rho(1-\rho)n\epsilon}. \tag{8}$$

Next we would like to estimate the right-hand side of (8). Due to the limitation of space, we shall only consider a simple truncation estimation, which leads to the correct convergence rate for non-parametric problems but yields an unnecessary $\ln n$ factor for parametric problems (which can be correctly handled with a more precise estimation). We introduce the following notation, which is essentially the prior measure of an $\epsilon$-radius $\mathbf{KL}$-ball around $q$:

$$M_\pi^{\mathbf{KL}}(\epsilon) = \pi(\mathbf{KL}(q||p(\cdot|\theta)) \leq \epsilon) = \mathbf{E}_\pi \mathbf{1}(\mathbf{KL}(q||p(\cdot|\theta)) \leq \epsilon).$$

Using this definition, we have $\mathbf{E}_\pi e^{-\alpha n \mathbf{KL}(q||p(\cdot|\theta))} \geq M_\pi^{\mathbf{KL}}(\epsilon)e^{-\alpha n\epsilon}$. In addition, we shall define the $\epsilon$-upper bracketing of $\Gamma$ (introduced in [1]), denoted by $N(\Gamma, \epsilon)$, as the minimum number of non-negative functions $\{f_i\}$ on $\mathcal{X}$ with respect to $\lambda$ such that $\mathbf{E}_q(f_i/q) = 1 + \epsilon$, and $\forall \theta \in \Gamma$, $\exists i$ such that $p(x|\theta) \leq f_i(x)$ a.e. $[\lambda]$. We have

$$\frac{1}{n}\mathbf{E}_X \sup_\theta \sum_{i=1}^n \ln \frac{p(X_i|\theta)}{q(X_i)} \leq \frac{1}{n}\mathbf{E}_X \ln \sum_{j=1}^{N(\Gamma,\epsilon)} e^{\sum_{i=1}^n \ln \frac{f_j(X_i)}{q(X_i)}}$$

$$\leq \frac{1}{n}\ln \sum_{j=1}^{N(\Gamma,\epsilon)} \mathbf{E}_X e^{\sum_{i=1}^n \ln \frac{f_j(X_i)}{q(X_i)}} = \frac{\ln N(\Gamma, \epsilon)}{n} + \ln(1 + \epsilon).$$

Therefore we obtain from (8) that $\forall s > 0$:

$$\rho(1 - \rho)s\mathbf{E}_X m_\pi^{\alpha,\rho}(X, s\epsilon) \leq \alpha - \frac{1}{n\epsilon}\ln M_\pi^{\mathbf{KL}}(\epsilon) + (\alpha - \rho)\frac{\ln N(\Gamma, \epsilon) + n\epsilon}{n\epsilon}.$$

The above bound immediately implies the following consistency and convergence rate theorem for Bayesian posterior distribution:

**Theorem 4.1** *Consider a sequence of Bayesian prior distributions $\pi_n$ on a parameter space $\Gamma_n$, which may be different for different sample sizes. Consider a sequence of positive numbers $\{\epsilon_n\}$ such that*

$$\sup_n \frac{-1}{n\epsilon_n}\ln M_{\pi_n}^{\mathbf{KL}}(\epsilon_n) < \infty, \tag{9}$$

*then $\forall s_n > 0$ such that $s_n \to \infty$, and $\forall \alpha \in (0, 1)$, $m_{\pi_n}^{\alpha,\alpha}(X, s_n\epsilon_n) \to 0$ in probability.*

*Moreover, if*

$$\sup_n \frac{\ln N(\Gamma_n, \epsilon_n)}{n\epsilon_n} < \infty, \tag{10}$$

*then $\forall s_n > 0$ such that $s_n \to \infty$, and $\forall \rho \in (0, 1)$, $m_{\pi_n}^{1,\rho}(X, s_n\epsilon_n) \to 0$ in probability.*

The first claim implies that for all $\alpha < 1$, the $\alpha$-Bayesian posterior $\pi^\alpha$ is concentrated in an $\epsilon_n$ ball around $q$ in $D_\alpha$ divergence, and the rate of convergence is $O_p(\epsilon_n)$. Note that $\epsilon_n$ is determined only by the local property of $\pi_n$ around the true distribution $q$. It also immediately implies that as long as $M_{\pi_n}^{\mathbf{KL}}(\epsilon) > 0$ for all $\epsilon > 0$, the $\alpha$-Bayesian method with $\alpha < 1$ is consistent.

The second claim applies to the standard Bayesian method. Its consistency requires an additional assumption (10), which depends on global properties of the prior $\pi_n$. This may seem somewhat surprising at first, but the condition is necessary. In fact, the counter-example given in [1] shows that the standard Bayesian method can be inconsistent even under the condition $M_{\pi_n}^{\mathbf{KL}}(\epsilon) > 0$ for all $\epsilon > 0$. Therefore a standard Bayesian procedure can be ill-behaved even if we put a sufficient amount of prior around the true distribution.

The consistency theorem given in [1] also relies on the upper entropy number $N(\Gamma, \epsilon)$. However, no convergence rates were established. Here we obtained a rate of convergence result for the standard Bayesian method using their covering definitions. Other definitions of covering (e.g. Hellinger covering) were used in more recent works to obtain rate of convergence for non-parametric Bayesian methods [3, 8]. Although it is possible to derive bounds using those different covering definitions in our analysis, we shall not work out the details here. However, we shall point out that these works made assumptions not completely necessary. For example, in [3], the definition of $M_\pi^{\mathbf{KL}}(\epsilon)$ requires additional assumptions that $\mathbf{E}_q \ln(q/p(\cdot|\theta))^2 \leq \epsilon^2$. This stronger condition is not needed in our analysis. Finally we shall mention that the bound of the form in Theorem 4.1 is known to produce optimal convergence rates for non-parametric problems (see [3, 8] for examples).

# 5  Conclusion

In this paper, we formulated an extended family of Bayesian algorithms as empirical log-risk minimization under entropy regularization. We then derived general posterior averaging bounds under entropy regularization that are suitable for analyzing Bayesian methods. These new bounds are of independent interests since they lead to Bernstein style exponential inequalities, which are crucial for obtaining the correct convergence behavior for many statistical estimation problems such as least squares regression.

Using the posterior averaging bounds, we obtain new convergence results for a generalized family of Bayesian posterior distributions. Our results imply that the $\alpha$-Bayesian method with $\alpha < 1$ is more robust than the standard Bayesian method since its convergence behavior is completely determined by the local prior density around the true distribution. Although the standard Bayesian method is "optimal" in a certain averaging sense, its behavior is heavily dependent on the regularity of the prior distribution globally. What happens is that the standard Bayesian method can put too much emphasis on the difficult part of the prior distribution, which degrades the estimation quality in the easier parts where we are actually more interested in. Therefore even if one is able to guess the true distribution by putting a large prior mass around its neighborhood, the Bayesian method can still ill-behave if one accidentally makes bad choices elsewhere. It is thus difficult to design good Bayesian priors. The new theoretical insights obtained here imply that unless one completely understands the impact of the prior, it is much safer to use an $\alpha$-Bayesian method.

## Acknowledgments

The author would like to thank Andrew Barron, Ron Meir, and Matthias Seeger for helpful discussions and comments.

## Footnotes

[1] In this paper, we view the Bayesian paradigm as a method to generate statistical inferencing procedures, and thus don't assume that the Bayesian prior assumption has to be true. In particular, we do not even assume that $q \in \{p(\cdot|\theta) : \theta \in \Gamma\}$.

[2]In this case, slightly tighter results can be obtained by applying the Hoeffding's exponential inequality directly to the left-hand side of Theorem 3.1, instead of the method used in Corollary 3.1.

## References

[1] Andrew Barron, Mark J. Schervish, and Larry Wasserman. The consistency of posterior distributions in nonparametric problems. *Ann. Statist.*, 27(2):536–561, 1999.

[2] Persi Diaconis and David Freedman. On the consistency of Bayes estimates. *Ann. Statist.*, 14(1):1–67, 1986. With a discussion and a rejoinder by the authors.

[3] Subhashis Ghosal, Jayanta K. Ghosh, and Aad W. van der Vaart. Convergence rates of posterior distributions. *Ann. Statist.*, 28(2):500–531, 2000.

[4] D. McAllester. PAC-Bayesian stochastic model selection. *Machine Learning*, 51(1):5–21, 2003.

[5] Ron Meir and Tong Zhang. Generalization error bounds for Bayesian mixture algorithms. *Journal of Machine Learning Research*, 4:839–860, 2003.

[6] C. P. Robert. *The Bayesian Choice: A Decision Theoretic Motivation*. Springer Verlag, New York, 1994.

[7] M. Seeger. PAC-Bayesian generalization error bounds for Gaussian process classification. *JMLR*, 3:233–269, 2002.

[8] Xiaotong Shen and Larry Wasserman. Rates of convergence of posterior distributions. *Ann. Statist.*, 29(3):687–714, 2001.
